# Locating Changes in Highly Dependent Data with Unknown Number of Change Points

**Azadeh Khaleghi**
SequeL-INRIA/LIFL-CNRS,
Université de Lille, France
azadeh.khaleghi@inria.fr

**Daniil Ryabko**
SequeL-INRIA/LIFL-CNRS,
daniil@ryabko.net

## Abstract

The problem of multiple change point estimation is considered for sequences with unknown number of change points. A consistency framework is suggested that is suitable for highly dependent time-series, and an asymptotically consistent algorithm is proposed. In order for the consistency to be established the only assumption required is that the data is generated by stationary ergodic time-series distributions. No modeling, independence or parametric assumptions are made; the data are allowed to be dependent and the dependence can be of arbitrary form. The theoretical results are complemented with experimental evaluations.

## 1   Introduction

We are given a sequence $\mathbf{x} := X_1, X_2, \ldots, X_n$ formed as the concatenation of an *unknown* number $k + 1$ of sequences, such that $\mathbf{x} = X_1 \ldots X_{\pi_1} X_{\pi_1+1} \ldots X_{\pi_2} \ldots X_{\pi_k} \ldots X_n$. The time-series distributions that generate a pair of adjacent sequences separated by indices $\pi_i$, $i = 1..k$ are different. (Non-adjacent sequences are allowed to be generated by the same distribution). The so-called *change points* $\pi_i$, $i = 1..k$ are *unknown* and to be estimated. Change point estimation is one of the core problems in statistics, and as such, has been studied extensively under various formulations. However, even nonparametric formulations of the problem typically assume that the data in each segment are independent and identically distributed, and that the change necessarily affects single-dimensional marginal distributions. In this paper we consider the most general nonparametric setting where, the changes may be completely arbitrary (e.g., in the form of the long-range dependence). We propose a change point estimation algorithm that is asymptotically consistent under such minimal assumptions.

**Motivation.** Change point analysis is an indispensable tool in a broad range of applications such as market analysis, bioinformatics, network traffic, audio/video segmentation only to name a few. Clearly, in these applications the data can be highly dependent and can not be easily modeled by parametric families of distributions. From a machine learning perspective, change point estimation is a difficult unsupervised learning problem: the objective is to estimate the change points in a given sequence while no labeled examples are available. To better understand the challenging nature of the problem, it is useful to compare it to time-series clustering. In time-series clustering, a *set of sequences* is to be partitioned, whereas in change point estimation the partitioning is done on a *sequence of sequences*. While objectives are the same, in the latter, information about the individual elements is no longer available, since only a single sequence formed by their concatenation is provided as input. This makes change point estimation a more challenging problem than time-series clustering.

In the general setting of highly-dependent time-series correct estimation of the number of change points is provably impossible, even in the weakest asymptotic sense, and even if there is at most one change [23]. While a popular mitigation is to consider more restrictive settings, we are interested in intermediate formulations that can have asymptotically consistent solutions under the most

general assumptions. In light of the similarities between clustering and change point analysis, we propose a formulation that is motivated by hierarchical clustering. When the number of clusters is unknown, a hierarchical clustering algorithm produces a tree, such that some pruning of this tree gives the ground-truth clustering (e.g., [3]). In change point estimation with an unknown number $k$ of change points, we suggest to aim for a *sorted list* of change points, whose first $k$ elements are some permutation of the true change points. An algorithm that achieves this goal is called consistent.

**Related Work.** Change point analysis is a classical problem in mathematical statistics [6, 4, 5, 17]. In a typical formulation, samples within each segment are assumed to be i.i.d. and the change usually refers to the change in the mean. More general formulations are often considered as well, however, it is usually assumed that the samples are i.i.d. in each of the segments [20, 8, 9, 21] or that they belong to some specific model class (such as Hidden Markov processes) [15, 16, 27]. In these frameworks the problem of estimating the number of change points is usually addressed with penalized criteria, see, for example, [19, 18]. In nonparametric settings, the typical assumptions usually impose restrictions on the form of the change or the nature of dependence (e.g., the time-series are assumed strongly mixing) [6, 4, 10, 12]. Even when more general settings are considered, it is almost exclusively assumed that the single-dimensional marginal distributions are different [7].

The framework considered in this paper is similar to that of [25] and of our recent paper [13], in the sense that the only assumption made is that the distributions generating the data are stationary ergodic. The particular case of $k = 1$ is considered in [25]. In [13] we provide a non-trivial extension of [25] for the case where $k > 1$ is known and is provided to the algorithm. However, as mentioned above, when the number $k$ of change points is unknown, it is provably impossible to estimate it, even under the assumption $k \in \{0, 1\}$ [23]. In particular, if the input $k$ is not the correct number of change points, then the behavior of the algorithm proposed in [13] can be arbitrary bad.

**Results.** We present a nonparametric change point estimation algorithm for time-series data with unknown number of change points. We consider the most general framework where the only assumption made is that the unknown distributions generating the data are stationary ergodic. This means that we make no such assumptions as independence, finite memory or mixing. Moreover, we do not need the finite-dimensional marginals of any fixed size before and after the change points to be different. Also, the marginal distributions are not required to have densities.

We show that the proposed algorithm is asymptotically consistent in the sense that among the change point estimates that it outputs, the first $k$ converge to the true change points. Moreover, our algorithm can be efficiently calculated; it has a computational complexity $\mathcal{O}(n^2 \text{ polylog } n)$ where $n$ is the length of the input sequence. To the best of our knowledge, this work is the first to address the change point problem with an unknown number of change points in such general framework.

We further confirm our theoretical findings through experiments on synthetic data. Our experimental setup is designed so as to demonstrate the generality of the suggested framework. To this end, we generate our data by time-series distributions that, while being stationary ergodic, do not belong to any "simpler" class of processes. In particular they cannot be modeled as hidden Markov processes with finite or countably infinite set of states. Through our experiments we show that the algorithm is consistent in the sense that as the length of the input sequence grows, the produced change point estimations converge to the actual change points.

**Organization.** In Section 2 we introduce some preliminary notation and definitions. We formulate the problem in Section 3. Section 4 presents our main theoretical results, including the proposed algorithm, and an informal description of how and why it works. In Section 5 we prove that the proposed algorithm is asymptotically consistent under the general framework considered; we also show that our algorithm can be computed efficiently. In Section 6 we present some experimental results, and finally in Section 7 we provide some concluding remarks and future directions.

## 2   Notation and definitions

Let $\mathcal{X}$ be some measurable space (the domain); in this work we let $\mathcal{X} = \mathbb{R}$, but extensions to more general spaces are straightforward. For a sequence $X_1, \ldots, X_n$ we use the abbreviation $X_{1..n}$. Consider the Borel $\sigma$-algebra $\mathcal{B}$ on $\mathcal{X}^\infty$ generated by the cylinders $\{B \times \mathcal{X}^\infty : B \in B^{m,l}, m, l \in \mathbb{N}\}$ where, the sets $B^{m,l}, m, l \in \mathbb{N}$ are obtained via the partitioning of $\mathcal{X}^m$ into cubes of dimension $m$ and volume $2^{-ml}$ (starting at the origin). Let also $B^m := \cup_{l \in \mathbb{N}} B^{m,l}$. Processes are probability

measures on the space $(\mathcal{X}^\infty, \mathcal{B})$. For $\mathbf{x} = X_{1..n} \in \mathcal{X}^n$ and $B \in B^m$ let $\nu(\mathbf{x}, B)$ denote the *frequency* with which $\mathbf{x}$ falls in $B$, i.e.

$$\nu(\mathbf{x}, B) := \frac{\mathbb{I}\{n \geq m\}}{n - m + 1} \sum_{i=1}^{n-m+1} \mathbb{I}\{X_{i..i+m-1} \in B\} \tag{1}$$

A process $\rho$ is *stationary* if for any $i, j \in 1..n$ and $B \in B^m$, $m \in \mathbb{N}$, we have $\rho(X_{1..j} \in B) = \rho(X_{i..i+j-1} \in B)$. A stationary process $\rho$ is called *(stationary) ergodic* if for all $B \in \mathcal{B}$ we have $\lim_{n \to \infty} \nu(X_{1..n}, B) = \rho(B)$ with $\rho$-probability 1. The *distributional distance* between a pair of process distributions $\rho_1$ and $\rho_2$ is defined as follows

$$d(\rho_1, \rho_2) := \sum_{m,l=1}^{\infty} w_m w_l \sum_{B \in B^{m,l}} |\rho_1(B) - \rho_2(B)|$$

where, $w_i := 2^{-i}$, $i \in \mathbb{N}$. Note that any summable sequence of positive scores also works. It is easy to see that $d(\cdot, \cdot)$ is a metric. For more on the distributional distance and its properties see [11].

In this work we use *empirical estimates* of this distance. Specifically, the empirical estimate of the distance between a sequence $\mathbf{x} = X_{1..n} \in \mathcal{X}^n$, $n \in \mathbb{N}$ and a process distribution $\rho$ is defined as

$$\hat{d}(\mathbf{x}, \rho) := \sum_{m,l=1}^{\infty} w_m w_l \sum_{B \in B^{m,l}} |\nu(\mathbf{x}, B) - \rho(B)| \tag{2}$$

and for a pair of sequences $\mathbf{x}_i \in \mathcal{X}^{n_i}$ $n_i \in \mathbb{N}$, $i = 1, 2$. it is defined as

$$\hat{d}(\mathbf{x}_1, \mathbf{x}_2) := \sum_{m,l=1}^{\infty} w_m w_l \sum_{B \in B^{m,l}} |\nu(\mathbf{x}_1, B) - \nu(\mathbf{x}_2, B)|. \tag{3}$$

Although expressions (2) and (3) involve infinite sums they can be easily calculated [22]. Moreover, the estimates $\hat{d}(\cdot, \cdot)$ are asymptotically consistent [25]: for any pair of stationary ergodic distributions $\rho_1, \rho_2$ generating sequences $\mathbf{x}_i \in \mathcal{X}^{n_i}$ $i = 1, 2$ we have

$$\lim_{n_1, n_2 \to \infty} \hat{d}(\mathbf{x}_1, \mathbf{x}_2) = d(\rho_1, \rho_2), \text{ a.s., and} \tag{4}$$

$$\lim_{n_i \to \infty} \hat{d}(\mathbf{x}_i, \rho_j) = d(\rho_i, \rho_j), \; i, j \in 1, 2, \text{ a.s.} \tag{5}$$

Moreover, a more general estimate of $(\cdot, \cdot)$ may be obtained as

$$\check{d}(\mathbf{x}_1, \mathbf{x}_2) := \sum_{m=1}^{m_n} \sum_{l=1}^{l_n} w_m w_l \sum_{B \in B^{m,l}} |\nu(\mathbf{x}_1, B) - \nu(\mathbf{x}_2, B)| \tag{6}$$

where, $m_n$ and $l_n$ are any sequences of integers that go to infinity with $n$. As shown in [22] the consistency results for $\hat{d}$, i.e. (2) and (3) equally hold for $\check{d}$ so long as $m_n$, $l_n$ go to infinity with $n$. Let $\mathbf{x} = X_{1..n}$ be a sequence and consider a subsequence $X_{a..b}$ of $\mathbf{x}$ with $a < b \in 1..n$. We define the intra-subsequence distance of $X_{a..b}$ as

$$\Delta_{\mathbf{x}}(a, b) := \hat{d}(X_{a..\lfloor \frac{a+b}{2} \rfloor}, X_{\lceil \frac{a+b}{2} \rceil..b}) \tag{7}$$

We further define the single-change point estimator of $X_{a..b}$, $a < b$ as

$$\Phi_{\mathbf{x}}(a, b, \alpha) := \underset{t \in [a,b]}{\operatorname{argmax}} \; \hat{d}(X_{a-n\alpha..t}, X_{t..b+n\alpha}), \; \alpha \in (0, 1) \tag{8}$$

## 3 Problem Formulation

We formalize the multiple change point estimation problem as follows. We are given a sequence

$$\mathbf{x} := X_1, \ldots, X_n \in \mathcal{X}^n$$

which is the concatenation of an *unknown* number $\kappa + 1$ of sequences

$$X_{1..\pi_1}, X_{\pi_1+1..\pi_2}, \ldots, X_{\pi_\kappa+1..n}.$$

Each of these sequences is generated by an *unknown stationary ergodic* process distribution. Moreover, every two consecutive sequences are generated by *two different* process distributions. (A pair of *non-consecutive* sequences may be generated by the same distribution.) The process distributions are not required to be independent. The parameters $\pi_k$ are *unknown* and have to be estimated; they are called *change points*. Note that it is not required for the means, variances or single-dimensional marginals of the distributions to be different. We are considering the most general scenario where the process distributions are different.

**Definition 1** (change point estimator). *A change point estimator is a function that takes a sequence* $\mathbf{x}$ *and a parameter* $\lambda \in (0,1)$ *and outputs a sequence of change point estimates,* $\hat{\boldsymbol{\pi}} := \hat{\pi}_1, \hat{\pi}_2, \ldots \hat{\pi}_{1/\lambda}$. *(Note that the total number of estimated change points* $1/\lambda$ *may be larger than the true number of change points* $\kappa$.)

To construct consistent algorithms, we assume that the *change points* $\pi_k$ *are linear in* $n$ i.e. $\pi_k := n\theta_k$ where $\theta_k \in (0,1)$ $k = 1..\kappa$ are *unknown*. We also define the minimum normalized distance between the change points as

$$\lambda_{\min} := \min_{k=1..\kappa+1} \theta_k - \theta_{k-1} \tag{9}$$

where $\theta_0 := 0$ and $\theta_{\kappa+1} := 1$, and assume $\lambda_{\min} > 0$. The reason why we impose these conditions is that the consistency properties we are after are asymptotic in $n$. If the length of one of the sequences is constant or sublinear in $n$ then asymptotic consistency is impossible in this setting. We define the consistency of a change point estimator as follows.

**Definition 2** (Consistency of a change point estimator). *Let* $\hat{\boldsymbol{\pi}} := \hat{\pi}_1, \hat{\pi}_2, \ldots \hat{\pi}_{1/\lambda}$ *be a change point estimator. Let* $\hat{\boldsymbol{\theta}}(\kappa) = (\hat{\theta}_1, \ldots, \hat{\theta}_\kappa) := \textbf{sort}(\frac{1}{n}\hat{\pi}_1, \ldots, \frac{1}{n}\hat{\pi}_\kappa)$, *where* **sort**$(\cdot)$ *orders the first* $\kappa$ *elements* $\hat{\pi}_1, \ldots, \hat{\pi}_\kappa$ *of* $\hat{\boldsymbol{\pi}}$ *with respect to their order of appearance in* $\mathbf{x}$. *We call the change point estimator* $\hat{\boldsymbol{\pi}}$ *asymptotically consistent if with probability 1 we have*

$$\lim_{n \to \infty} \sup_{k=1..\kappa} |\hat{\theta}_k - \theta_k| = 0.$$

## 4 Theoretical Results

In this section we introduce a nonparametric multiple change point estimation algorithm for the case where the number of change points is unknown. We also give an informal description of the algorithm, and intuitively explain why it works. The main result is Theorem 1 which states that the proposed algorithm is consistent under the most general assumptions. Moreover, the computational complexity of the algorithm is $\mathcal{O}(n^2 \, \text{polylog} \, n)$ where $n$ denotes the length of the input sequence.

The main steps of the algorithm are as follows. Given $\lambda \in (0,1)$, a sequence of evenly-spaced indices is formed. The index-sequence is used to partition $\mathbf{x} = X_{1..n}$ into consecutive segments of length $n\alpha$, where $\alpha := \frac{\lambda}{3}$. The single-change point estimator $\Phi(\cdot, \cdot, \cdot)$ is used to generate a candidate change point within every segment. Moreover, the intra-subsequence-distance $\Delta(\cdot, \cdot)$ of each segment is used as its performance score $s(\cdot, \cdot)$. The change point candidates are ordered according to the performance-scores of their corresponding segments. The algorithm assumes the input parameter $\lambda$ to be a lower-bound on the true normalized minimum distance $\lambda_{\min}$ between actual change points. Hence, the sorted list of estimated change points is filtered in such a way that its elements are at least $\lambda n$ apart. The algorithm outputs an *ordered* sequence $\hat{\boldsymbol{\pi}}$ of change point estimates, where the ordering is done with respect to the performance scores $s(\cdot, \cdot)$. The length of $\hat{\boldsymbol{\pi}}$ may be larger than $\kappa$. However, as we show in Theorem 1, from some $n$ on, the first $\kappa$ elements $\hat{\pi}_k$, $k = 1..\kappa$ of the output $\hat{\boldsymbol{\pi}}$ converge to some permutation of the true change points, $\pi_1, \cdots, \pi_\kappa$.

**Theorem 1.** *Let* $\mathbf{x} := X_{1..n} \in \mathcal{X}^n$, $n \in \mathbb{N}$ *be a sequence with change points at least* $n\lambda_{\min}$ *apart, for some* $\lambda_{\min} \in (0,1)$. *Then Alg1$(\mathbf{x}, \lambda)$ is asymptotically consistent for* $\lambda \in (0, \lambda_{\min}]$.

**Remark 2** (Computational complexity). While the definition (3) of $\hat{d}(\cdot, \cdot)$ involves taking infinite sums, the distance can be calculated efficiently. Indeed, in (3) all summands corresponding to $m > \max_{i=1,2} n_i$ equal 0; moreover, all summands corresponding to $l > s_{\min}$ are equal, where $s_{\min}$ corresponds to the partition in which each cell has at most one point in it $s_{\min} := \min_{i,j \in 1..n, \, X_i \neq X_j} |X_i - X_j|$. Thus, even with a most naive implementation the computational complexity of the algorithm is at most polynomial in all arguments. A more efficient implementation can be obtained if one uses $\check{d}(\cdot, \cdot)$ given by (6), instead of $\hat{d}(\cdot, \cdot)$, with $m = \log n$,

---
**Algorithm 1** Estimating the change points
---

**input:** Sequence $\mathbf{x} = X_{1..n}$, Minimum Normalized Distance between the change points $\lambda$
**initialize:** Step size $\alpha \leftarrow \frac{\lambda}{3}$, Output change point Sequence $\hat{\boldsymbol{\pi}} \leftarrow ()$
**1. Generate** $2$ **sets of index-sequences:**

$$b_i^t \leftarrow n\alpha(i + \frac{1}{t+1}),\ i = 0..\frac{1}{\alpha},\ t = 1, 2$$

**2. Calculate the intra-distance value (given by** (7)**) of every segment** $X_{b_i^t..b_{i+1}^t}$, $i = 1..\frac{1}{\alpha}, t = 1, 2$ **as its performance score:**

$$s(t, i) \leftarrow \Delta_\mathbf{x}(b_i^t, b_{i+1}^t),\ i = 1..\frac{1}{\alpha}, t = 1, 2$$

**3. Use the single-change point-estimator (given by** (8)**) to estimate a change point in every segment:**

$$\hat{p}(t, i) := \Phi_\mathbf{x}(b_i^t, b_{i+1}^t, \alpha),\ i = 1..\frac{1}{\alpha} - 1,\ t = 1, 2$$

**4. Remove duplicates and sort based on scores:**

$$\mathcal{U} \leftarrow \{(t, i) : i \in 1..\frac{1}{\alpha} - 1, t = 1, 2\}$$

**while** $\mathcal{U} \neq \varnothing$ **do**

  **i. Select an available change point estimate of highest score and add it to** $\hat{\pi}$**:**

$$(\tau, l) \leftarrow \mathrm{argmax}_{(t,i) \in \mathcal{U}}\ s(t, i)\ \text{- break the ties arbitrarily}$$
$$\hat{\boldsymbol{\pi}} \leftarrow \hat{\boldsymbol{\pi}} \oplus \hat{p}(\tau, l),\ \text{i.e. append } \hat{\boldsymbol{\pi}} \text{ with } \hat{p}(\tau, l)$$

  **ii. Remove the estimates within a radius of** $\lambda n/2$ **from** $\hat{\pi}(l)$**:**

$$\mathcal{U} \leftarrow \mathcal{U} \setminus \{(t, i) : \hat{p}(t, i) \in (\hat{p}(\tau, l) - \lambda n/2, \hat{p}(\tau, l) + \lambda n/2)\}$$

**end while**
**return:** A sequence $\hat{\boldsymbol{\pi}}$ of change point estimates. Note: Elements of $\hat{\boldsymbol{\pi}}$ are at least $n\lambda$ apart and are sorted in decreasing order of their scores $\mathbf{s}(\cdot, \cdot)$.

---

where $n$ is the length of the samples; in this case, the consistency results are unaffected, and the computational complexity of calculating the distance becomes $n\ \mathrm{polylog}\ n$, making the complexity of the algorithm $n^2\ \mathrm{polylog}\ n$. The choice $m = \log n$ is further justified by the fact that the frequencies of cells in $B^{m,l}$ corresponding to higher values of $m$ are not consistent estimates of their probabilities (and thus only add to the error of the estimate); see [22, 14] for further discussion.

The proof of the theorem is given in the next section. Here we provide an intuition as to why the consistency statement holds.
First, recall that the empirical distributional distance between a given pair of sequences converges to the distributional distance between the corresponding process distributions. Consider a sequence $\mathbf{x} = X_{1..n}$, and assume that a segment $X_{a..b}$, $a, b \in 1..n$ does not contain any change points, so that $X_{a..\frac{a+b}{2}}$ and $X_{\frac{a+b}{2}..b}$ are generated by the same process. If the length of $X_{a..b}$ is linear in $n$, so that $b - a = \alpha n$ for some $\alpha \in (0, 1)$, then its intra-subsequence distance $\Delta_\mathbf{x}(a, b)$ (defined by (7)) converges to $0$ with $n$ going to infinity. On the other hand, if there is a single change point $\pi$ within $X_{a..b}$ whose distance from $a$ and $b$ is linear in $n$, then $\Delta_\mathbf{x}(a, b)$ converges to a non-zero constant.

Now assume that $X_{a..b}$ with its change point at $\pi \in a..b$ is contained within a larger segment $X_{a-n\alpha'..b+n\alpha'}$ for some $\alpha' \in (0, 1)$. In this case, the single-change point estimator $\Phi(a, b, \alpha')$ (defined by (8)) produces an estimate that from some $n$ on converges to $\pi$ provided that $\pi$ is the only change point in $X_{a-n\alpha'..b+n\alpha'}$. These observations are key to the consistency of the algorithm.

When $\lambda \leq \lambda_{\min}$, each of the index-sequences generated with $\alpha := \frac{\lambda}{3}$ partitions $\mathbf{x}$ in such a way that every three consecutive segments of the partition contain *at most* one change point. Also, the segments are of lengths linear in $n$. In this scenario, from some $n$ on, the change point estimator $\Phi(\cdot, \cdot, \cdot)$ produces correct candidates within each of the segments that contains a true change point. Moreover, from some $n$ on, the performance scores $s(\cdot, \cdot)$ of the segments without change points converge to $0$, while those corresponding to the segments that encompass a change point converge

to a non-zero constant. Thus from some $n$ on, the $\kappa$ change point candidates of highest performance score that are at least at a distance $\lambda n$ from one another, each converge to a unique change point.

A problem occurs if the generated index-sequence is such that it includes some of the change points as elements. As a mitigation strategy, we generate two index-sequences with the same gap $\alpha n$ between their consecutive elements but with distinct starting points: one starts at $\frac{n\alpha}{2}$ and the other at $\frac{n\alpha}{3}$. Each index-sequence gives a different partitioning of $\mathbf{x}$ into consecutive segments. This way, every change point is fully encompassed by *at least* one segment from either of the two partitions. We choose the appropriate segments based on their performance scores. From the above argument we can see that segments with change points will have higher scores, and the change points within will be estimated correctly; finally, this is used to prove the theorem in the next session.

## 5 Proof of Theorem 1

*The proof relies on Lemma 1 and Lemma 2, which we borrow from [13] and state here without proof. We also require the following additional notation.*

**Definition 3.** *For every change point $\pi_k$, $k = 1..\kappa$ and every fixed $t = 1,2$ we denote by $L^t(\pi_k)$ and by $R^t(\pi_k)$ the elements of the index-sequence $b_i^t$, $i = 1..\frac{1}{\alpha}$ that appear immediately to the left and to the right of $\pi_k$ respectively, i.e. $L^t(\pi_k) := \max\limits_{b_i^t \leq \pi_k,\ i=0..\frac{1}{\alpha}} b_i^t$ and $R^t(\pi_k) := \min\limits_{b_i^t \geq \pi_k,\ i=0..\frac{1}{\alpha}} b_i^t$.*
*(Equality occurs when $\pi_k$ for some $k \in 1..\kappa$ is exactly at the start or at the end of a segment.)*

**Lemma 1** ([13]). *Let $\mathbf{x} = X_{1..n}$ be generated by a stationary ergodic process $\rho$. For all $\zeta \in [0,1)$ and $\alpha \in (0,1)$ we have, $\lim\limits_{n\to\infty} \sup\limits_{b_1 \geq \zeta n,\ b_2 \geq b_1 + \alpha n} \Delta_{\mathbf{x}}(b_1, b_2) = 0$.*

**Lemma 2** ([13]). *Let $\delta$ denote the minimum distance between the distinct distributions generating the data. Denote by $\kappa$ the "unknown" number of change points and assume that for some $\zeta \in (0,1)$ and some $t = 1,2$ we have, $\inf\limits_{\substack{k=1..\kappa \\ i=0..\frac{1}{\alpha}}} |b_i^t - \pi_k| \geq \zeta n$.*
*(i) With probability one we have, $\lim\limits_{n\to\infty} \inf\limits_{k\in1..\kappa} \Delta_{\mathbf{x}}(L^t(\pi_k), R^t(\pi_k)) \geq \delta\zeta$.*
*(ii) If additionally we have that $[L^t(\pi_k) - n\alpha, R^t(\pi_k) + n\alpha] \subseteq [\pi_{k-1}, \pi_{k+1}]$ then with probability one we obtain, $\lim\limits_{n\to\infty} \sup\limits_{k\in1..\kappa} \frac{1}{n}|\Phi_{\mathbf{x}}(L^t(\pi_k), R^t(\pi_k), \alpha) - \pi_k| = 0$.*

*Proof of Theorem 1.* We first give an outline of the proof. In order for a change point $\pi_k$, $k \in 1..\kappa$ to be estimated correctly through this algorithm, there needs to be at least one $t = 1,2$ such that

**1.** $\pi_k \in (L^t(\pi_k), R^t(\pi_k))$ and **2.** $[L^t(\pi_k) - n\alpha, R^t(\pi_k) + n\alpha] \subseteq [\pi_{k-1}, \pi_k]$

where $\alpha := \frac{\lambda}{3}$, as specified by the algorithm. We show that from some $n$ on, for every change point the algorithm selects an appropriate segment satisfying these conditions, and assigns it a performance score $s(\cdot, \cdot)$ that converges to a non-zero constant. Moreover, the performance scores of the segments without change points converge to 0. Recall that, the change point candidates are finally sorted according to their performance scores, and the sorted list is filtered to include only elements that are at least $\lambda n$ apart. For $\lambda \leq \lambda_{\min}$, from some $n$ on, the first $\kappa$ elements of the output change point sequence $\hat{\boldsymbol{\pi}}$ are some permutation of the true change points. The proof follows.

Fix an $\varepsilon > 0$. Recall that the algorithm specifies $\alpha := \frac{\lambda}{3}$ and generates a sequence of evenly-spaced indicies $b_i^t := n\alpha(i + \frac{1}{t+1})$, $i = 1..\frac{1}{\alpha}$, $t = 1,2$. Observe that

$$b_i^t - b_{i-1}^t = n\alpha,\ i = 1..\frac{1}{\alpha}. \tag{10}$$

For every $i \in 0..\frac{1}{\alpha}$ and $t \in 1,2$ we have that the index $b_i^t$ is either exactly equal to a change point or has a linear distance from it. More formally, define $\zeta(t,i) := \min\limits_{k\in1..\kappa} |\alpha(i+\frac{1}{t+1}) - \theta_k|$, $i \in 0..1/\alpha\ t \in$ 1..2. (Note that $\zeta(t,i)$ can also be zero). For all $i \in 0..\frac{1}{\alpha}$, $t = 1,2$ and $k \in 1..\kappa$ we have

$$|b_i^t - \pi_k| \geq n\zeta(t,i). \tag{11}$$

For every $t = 1, 2$ and $i = 0..1/\alpha$, a performance score $s(t, i)$ is calculated as the intra-subsequence distance $\Delta_{\mathbf{x}}(b_i^t, b_{i+1}^t)$ of the segment $X_{b_i^t..b_{i+1}^t}$. Let $\mathcal{I} := \{(t, i) : t \in 1, 2, \ i \in 1..\frac{1}{\alpha} \text{ s.t. } \exists k \in 1..\kappa, \ \pi_k \in (b_i^t, b_{i+1}^t)\}$. Also define the complement set $\mathcal{I}' := \{1, 2\} \times \{1..\frac{1}{\alpha}\} \setminus \mathcal{I}$. By (10), (11) and Lemma 1, there exists some $N_1$ such that for all $n \geq N_1$ we have,

$$\sup_{(t,i) \in \mathcal{I}'} s(t, i) \leq \varepsilon. \tag{12}$$

Since $\lambda \leq \lambda_{\min}$, we have $\alpha \in (0, \lambda_{\min}/3]$. Therefore, for every $t = 1, 2$ and every change point $\pi_k, \ k \in 1..\kappa$ we have

$$[L^t(\pi_k) - n\alpha, R^t(\pi_k) + n\alpha] \subseteq [\pi_{k-1}, \pi_{k+1}]. \tag{13}$$

Define $\mu_{\min} := \min_{(t,i) \in \mathcal{I}} \zeta(t, i)$. It follows from the definition of $\mathcal{I}$ that

$$\mu_{\min} > 0. \tag{14}$$

By (10), (11), (13), (14) and Lemma 2.(i), there exists some $N_2$ such that for all $n \geq N_2$ we have

$$\inf_{(t,i) \in \mathcal{I}} s(t, i) \geq \delta\mu_{\min} \tag{15}$$

where $\delta$ denotes the minimum distance between the distributions. Let $\pi(t, i), \ i \in 0..1/\alpha, \ t = 1, 2$ denote the change point that is contained within $b_i^t..b_{i+1}^t$, $(t, i) \in \mathcal{I}$, i.e. $\pi(t, i) := \pi_k, \ k \in 1..\kappa$ s.t. $\pi_k \in (b_i^t, b_{i+1}^t)$. As specified in Step 3, the change point candidates are obtained as $\hat{p}(t, i) := \Phi_{\mathbf{x}}(b_i^{\tau(i)}, b_{i+1}^{\tau(i)}, \alpha), \ i = 1..1/\alpha - 1$. By (10), (11), (13), (14) and Lemma 2.(ii) there exists some $N_4$ such that for all $n \geq N_4$ we have

$$\sup_{(t,i) \in \mathcal{I}} \frac{1}{n} |\hat{p}(t, i) - \pi(t, i)| \leq \varepsilon. \tag{16}$$

Let $N := \max_{i=1..4} N_i$. Recall that (as specified in Step 4), the algorithm generates an output sequence $\hat{\boldsymbol{\pi}} := \hat{\pi}_1, \ldots, \hat{\pi}_{1/\lambda}$ by first sorting the change point candidates according to their performance scores, and then filtering the sorted list so that the remaining elements are at least $n\lambda$ apart. It remains to see that the corresponding estimate of every change point appears exactly once in $\hat{\boldsymbol{\pi}}$. By (12) and (15) for all $n \geq N$ the segments $b_i^t..b_{i+1}^t$, $(t, i) \in \mathcal{I}$ are assigned higher scores than $b_i^t..b_{i+1}^t$, $(t, i) \in \mathcal{I}'$. Moreover, by construction for every change point $\pi_k, \ k = 1..\kappa$ there exists some $(t, i) \in \mathcal{I}$ such that $\pi_k = \pi(t, i)$ which, by (16) is estimated correctly for all $n \geq N$. Next we show that every estimate appears *at most* once in the output sequence $\hat{\boldsymbol{\pi}}$. By (16) for all $(t, i), (t', i') \in \mathcal{I}$ such that $\pi(t, i) = \pi(t', i')$ and all $n \geq N$ we have

$$\frac{1}{n} |\hat{p}(t, i) - \hat{p}(t', i')| \leq \frac{1}{n} |\hat{p}(t, i) - \pi(t, i)| + \frac{1}{n} |\hat{p}(t', i') - \pi(t', i')| \leq 2\varepsilon. \tag{17}$$

On the other hand, for all $(t, i), (t', i') \in \mathcal{I}$ such that $\pi(t, i) \neq \pi(t', i')$ and all $n \geq N$ we have

$$\frac{1}{n} |\hat{p}(t, i) - \hat{p}(t', i')| \geq \frac{1}{n} |\pi(t, i) - \pi(t', i')| - \frac{1}{n} |\hat{p}(t, i) - \pi(t, i)| - \frac{1}{n} |\hat{p}(t', i') - \pi(t', i')|$$

$$\geq \frac{1}{n} |\pi(t, i) - \pi(t', i')| - 2\varepsilon \geq \lambda_{\min} - 2\varepsilon \tag{18}$$

where the last inequality follows from (16) and that the true change points are at least $n\lambda_{\min}$ apart. By (17) and (18) the duplicate estimates of every change point are filtered, while estimates corresponding to different change points are left untouched. Finally, following the notation of Definition 2, let $\hat{\boldsymbol{\theta}}(\kappa) = (\hat{\theta}_1, \ldots, \hat{\theta}_\kappa) := \mathbf{sort}(\frac{1}{n}\hat{\pi}_1, \cdots, \frac{1}{n}\hat{\pi}_\kappa)$, (sorted with respect to their order of appearance in $\mathbf{x}$). For $n \geq N$ we have, $\sup_{k \in 1..\kappa} |\hat{\theta}_k - \theta_k| \leq \varepsilon$ and the statement follows. $\qquad \square$

## 6  Experimental Results

In this section we use synthetically generated time-series data to empirically evaluate our algorithm. To generate the data we have selected distributions that while being stationary ergodic, do not belong to any "simpler" class of time-series, and are difficult to approximate by finite-state models. In particular they cannot be modeled by a hidden Markov process with a finite state-space. These distributions were used in [26] as examples of stationary ergodic processes which are not $B$-processes.

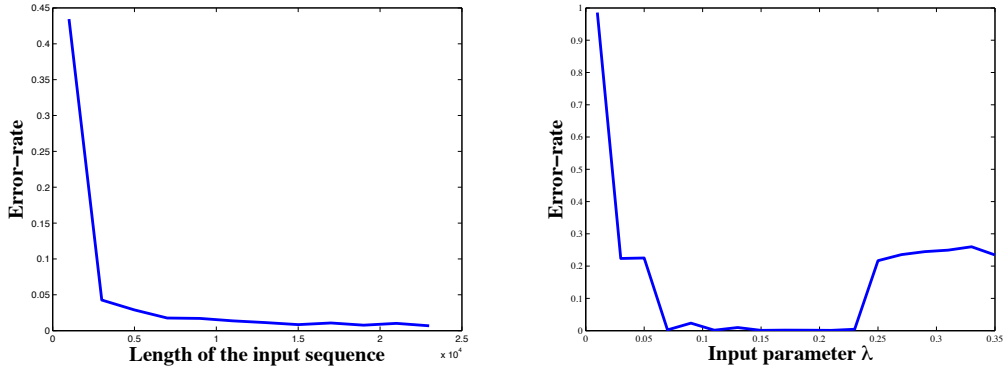

Figure 1: Left (Experiment 1): Average (over 20 runs) error as a function of the length of the input sequence. Right (Experiment 2): Average (over 25 runs) error as a function the input parameter $\lambda$.

**Time-series generation.** To generate a sequence $\mathbf{x} = X_{1..n}$ we proceed as follows. Fix some parameter $\alpha \in (0,1)$ and select $r_0 \in [0,1]$. For each $i = 1..n$ let $r_i = r_{i-1} + \alpha - \lfloor r_{i-1} + \alpha \rfloor$. The samples $X_i$ are obtained from $r_i$ by thresholding at 0.5, i.e. $X_i := \mathbb{I}\{r_i > 0.5\}$. We call this procedure $DAS(\alpha)$. If $\alpha$ is irrational then $\mathbf{x}$ forms a stationary ergodic time-series. We simulate $\alpha$ by a longdouble with a long mantisa. For the purpose of our experiments we use four different process distributions $DAS(\alpha_i)$, $i = 1..4$ with $\alpha_1 = 0.30...$, $\alpha_2 = 0.35...$, $\alpha_3 = 0.40...$ and $\alpha_4 = 0.45....$ To generate an input sequence $\mathbf{x} = X_{1..n}$ we fix some $\lambda_{\min} = 0.23$ and randomly generate $\kappa = 3$ change points at a minimum distance $n\lambda_{\min}$. We use $DAS(\alpha_i)$, $i = 1..4$ to respectively generate the four subsequences between every pair of consecutive change points.
**Experiment 1: (Convergence with Sequence Length)** In this experiment we demonstrate that the estimation error converges to 0 as the sequence length grows. We iterate over $n = 1000..20000$; at every iteration we generate an input sequence of length $n$ as described above. We apply Algorithm 1 with $\lambda = 0.18$ to find the change points. Figure 1 (Left) shows the average error-rate as a function of sequence length.
**Experiment 2: (Dependence on $\lambda$)** Algorithm 1 requires $\lambda \in (0,1)$ as a lower-bound on $\lambda_{\min}$. In this experiment we show that this lower bound need not be tight. In particular, there is a rather large range of $\lambda \leq \lambda_{\min}$ for which the estimation error is low. To demonstrate this, we fixed the sequence length $n = 20000$ and observed the error-rate as we varied the input parameter $\lambda$ between $0.01..0.35$. Figure 1 (Right) shows the average error-rate as a function of $\lambda$.

## 7 Outlook

In this work we propose a consistency framework for multiple change points estimation in highly dependent time-series, for the case where the number of change points is unknown. The notion of consistency that we consider requires an algorithm to produce a list of change points such that the first $k$ change points approach the true unknown change points in asymptotic. While in the general setting that we consider it is not possible to estimate the number of change points, other related formulations may be of interest. For example, if the number of different time-series distributions is known, but the number of change points is not, it may still be possible to estimate the latter. A simple example of this scenario would be when two distributions generate many segments in alternation.

While the consistency result here (and in the previous works [14, 22, 25]) rely on the convergence of frequencies, recent results of [1, 2] on uniform convergence can be used (see [24]) to solve related statistical problems about time-series (e.g., clustering) and thus may also prove useful in change point analysis.

**Acknowledgements.** This work is supported by the French Ministry of Higher Education and Research, Nord-Pas-de-Calais Regional Council and FEDER through CPER 2007-2013, ANR projects EXPLO-RA (ANR-08-COSI-004) and Lampada (ANR-09-EMER-007), by an INRIA Ph.D. grant to Azadeh Khaleghi, by the European Community's Seventh Framework Programme (FP7/2007-2013) under grant agreement 231495 (project CompLACS), and by Pascal-2.

# References

[1] Terrence M. Adams and Andrew B. Nobel. Uniform convergence of Vapnik-Chervonenkis classes under ergodic sampling. *The Annals of Probability*, 38:1345–1367, 2010.

[2] Terrence M. Adams and Andrew B. Nobel. Uniform approximation and bracketing properties of VC classes. *Bernoulli*, to appear.

[3] M.F. Balcan and P. Gupta. Robust hierarchical clustering. In *COLT*, 2010.

[4] M. Basseville and I.V. Nikiforov. *Detection of abrupt changes: theory and application*. Prentice Hall information and system sciences series. Prentice Hall, 1993.

[5] P.K. Bhattacharya. Some aspects of change-point analysis. *Lecture Notes-Monograph Series*, pages 28–56, 1994.

[6] B.E. Brodsky and B.S. Darkhovsky. *Nonparametric methods in change-point problems*. Mathematics and its applications. Kluwer Academic Publishers, 1993.

[7] E. Carlstein and S. Lele. Nonparametric change-point estimation for data from an ergodic sequence. *Teor. Veroyatnost. i Primenen.*, 38:910–917, 1993.

[8] L. Dumbgen. The asymptotic behavior of some nonparametric change-point estimators. *The Annals of Statistics*, 19(3):pp. 1471–1495, 1991.

[9] D. Ferger. Exponential and polynomial tailbounds for change-point estimators. *Journal of statistical planning and inference*, 92(1-2):73–109, 2001.

[10] L. Giraitis, R. Leipus, and D. Surgailis. The change-point problem for dependent observations. *Journal of Statistical Planning and Inference*, 53(3), 1996.

[11] R. Gray. *Prob. Random Processes, & Ergodic Properties*. Springer Verlag, 1988.

[12] S. B. Hariz, J. J. Wylie, and Q. Zhang. Optimal rate of convergence for nonparametric change-point estimators for nonstationary sequences. *Annals of Statistics*, 35(4):1802–1826, 2007.

[13] A. Khaleghi and D. Ryabko. Multiple change-point estimation in highly dependent time series. Technical report, arXiv:1203.1515, 2012.

[14] A. Khaleghi, D. Ryabko, J. Mary, and P. Preux. Online clustering of processes. In *AISTATS*, JMLR W&CP 22, pages 601–609, 2012.

[15] J. Kohlmorgen and S. Lemm. A dynamic hmm for on-line segmentation of sequential data. *Advances in Neural Inf. Proc. Systems*, 14:793–800, 2001.

[16] John D. Lafferty, Andrew McCallum, and Fernando C. N. Pereira. Conditional random fields: Probabilistic models for segmenting & labeling sequence data. In *ICML*, 2001.

[17] T.L. Lai. Sequential changepoint detection in quality control and dynamical systems. *Journal of the Royal Statistical Society*, pages 613–658, 1995.

[18] Marc Lavielle. Using penalized contrasts for the change-point problem. *Signal Processing*, 85(8):1501 – 1510, 2005.

[19] E. Lebarbier. Detecting multiple change-points in the mean of gaussian process by model selection. *Signal Processing*, 85(4):717 – 736, 2005.

[20] C.B. Lee. Nonparametric multiple change-point estimators. *Statistics & probability letters*, 27(4):295–304, 1996.

[21] Hidetoshi Murakami. A nonparametric locationscale statistic for detecting a change point. *The Inter. Journal of Advanced Manufacturing Technology*, 2001.

[22] D. Ryabko. Clustering processes. In *ICML*, pages 919–926, Haifa, Israel, 2010.

[23] D. Ryabko. Discrimination between B-processes is impossible. *Journal of Theoretical Probability*, 23(2):565–575, 2010.

[24] D. Ryabko and J. Mary. Reducing statistical time-series problems to binary classification. In *NIPS*, Lake Tahoe, USA, 2012.

[25] D. Ryabko and B. Ryabko. Nonparametric statistical inference for ergodic processes. *IEEE Transactions on Information Theory*, 56(3), 2010.

[26] P. Shields. *The Ergodic Theory of Discrete Sample Paths*. AMS Bookstore, 1996.

[27] X. Xuan and K. Murphy. Modeling changing dependency structure in multivariate time series. In *ICML*, pages 1055–1062. ACM, 2007.

